# Perceptual Metamers
# in Stereoscopic Vision

**Benjamin T. Backus** [*]
Department of Psychology
University of Pennsylvania
Philadelphia, PA 19104-6196
*backus@psych.upenn.edu*

## Abstract

Theories of cue combination suggest the possibility of constructing visual stimuli that evoke different patterns of neural activity in sensory areas of the brain, but that cannot be distinguished by any behavioral measure of perception. Such stimuli, if they exist, would be interesting for two reasons. First, one could know that none of the differences between the stimuli survive past the computations used to build the percepts. Second, it can be difficult to distinguish stimulus-driven components of measured neural activity from top-down components (such as those due to the interestingness of the stimuli). Changing the stimulus without changing the percept could be exploited to measure the stimulus-driven activity. Here we describe stimuli in which vertical and horizontal disparities trade during the construction of percepts of slanted surfaces, yielding stimulus equivalence classes. Equivalence class membership changed after a change of vergence eye posture alone, without changes to the retinal images. A formal correspondence can be drawn between these "perceptual metamers" and more familiar "sensory metamers" such as color metamers.

## 1   Introduction

Two types of perceptual process might, in principle, map physically different visual stimuli onto the same percept. First, the visual system has a host of constancy mechanisms that extract information about the visual environment across uninteresting changes in the proximal stimulus. Some of these mechanisms could be "leak-proof," leaving no trace of the original differences between the stimuli. Second, the visual system must combine information from redundant cues if it is to build percepts robustly. Recent cue conflict experiments have shown that the visual system's estimate of a scene parameter, as evinced in a visual percept, is often simply a weighted average of the parameter as specified by each cue separately [1]-[2]. Thus, a properly balanced cue-conflict stimulus might come to evoke the same percept as a "natural" or cue-concordant stimulus.

---

[*] http://psych.upenn.edu/~backus

Here, random-dot stereograms will be used to argue that leak-proof versions of both types of process exist. When a vertical magnifier is placed before one eye, a truly frontoparallel surface appears slanted. Adding horizontal magnification in the same eye restores frontoparallel appearance. The original stimulus and the magnified stimulus therefore have different patterns of binocular disparity but give rise to similar judgments of surface slant [3]. We show here that such stimuli are perceptually indistinguishable to practiced observers in a psychophysical discrimination task, which implies the loss of some disparity information.

This loss could occur, first, in a well-studied constancy mechanism that uses vertical disparity to correct the depth relief pattern associated with horizontal disparity [4]. However, the amount of horizontal magnification needed to null vertical magnification is less than would be predicted from use of this constancy mechanism alone; a second constancy mechanism exists that corrects horizontal disparities by using felt eye position, not vertical disparity [5]. Adding vertical magnification without changing eye position therefore creates a cue conflict stimulus. We show here that the amount of horizontal magnification needed to null the vertical magnification changes with the vergence posture of the eyes, which implies that both types of process (constancy and cue combination) are leak-proof across certain ranges of variation (magnifications) in these stereoscopic stimuli.

## 2 Stereoscopic slant perception: review of theory

The stereo component of the perceived slant of a random-dot surface can be modeled as the visual system's weighted average of two stereo slant estimates [5]-[6]. Horizontal disparity is ambiguous because it depends not only on surface slant, but also on surface patch location relative to the head. One stereo estimator resolves this ambiguity using vertical disparity (images are vertically larger in the closer eye), and the other resolves it using felt eye position. Vertical magnification in one eye thus creates a cue-conflict because it affects only the estimator that uses vertical disparity.

The two stereo estimators have different relative reliability at different distances, so the weights assigned to them by the visual system changes as a function of distance [7]. Since vergence eye posture is a cue to distance [8], one might predict that "perceptually metameric" stereo stimuli, if they exist, will lose their metameric status after a pure change of vergence eye posture that preserves the metamers' retinal images [9].

We shall now briefly describe the two stereoscopic slant estimators. This theory is covered elsewhere in greater detail [5]. Although surface slant has two components (slant and tilt [10]), we will consider only slant about a vertical axis. The arguments can be extended to slant about axes of arbitrary orientation [5].

The visual signals used in stereoscopic slant perception can be conveniently parameterized by four numbers [5]. Each can be considered a signal. A surface patch typically gives rise to all four signals. Two signals are the horizontal gradient of horizontal disparity, and the vertical gradient of vertical disparity, which we parameterize as horizontal size ratio (HSR) and vertical size ratio (VSR), respectively, in the manner of Rogers and Bradshaw [11]. They are defined as the horizontal (or vertical) size of the patch in the left eye, divided by the horizontal (or vertical) size in the right eye. These two signals must be measured from the retinal images. The two remaining signals are the headcentric azimuth and vergence of the surface patch. These signals can be known either by measuring the eyes' version and vergence, respectively, or from the retinal images [12].

A very good approximation that relates surface slant to horizontal disparity and VSR is:

$$S_{HSR,VSR} = -\tan^{-1}\left[\frac{1}{\mu} \ln \frac{HSR}{VSR}\right]$$                    **Equation 1**

where $\mu$ is the vergence of the surface patch in radians. We call this method of slant estimation *slant from HSR and VSR*.

A very good approximation that relates surface slant to horizontal disparity and azimuth is:

$$S_{HSR,EP} = -\tan^{-1}\left[\frac{1}{\mu} \ln HSR - \tan \gamma\right]$$                    **Equation 2**

where $\gamma$ is the azimuth of the surface patch. We call this method of slant estimation *slant from HSR and eye position* on the supposition that azimuth *per se* is known to the visual system primarily through measurement of the eyes' version.

Each estimator uses three of the four signals available to estimate surface slant from horizontal disparity. Nonstereo slant estimates can be rendered irrelevant by the choice of task, in which case perceived slant is a weighted average of the slants predicted from these two stereoscopic slant estimates [5, 6]. In principle, the reliability of slant estimation by HSR and eye position is limited at short viewing distances (large $\mu$) by error in the measurement of $\gamma$. Slant from HSR and VSR, on the other hand, continues to become more reliable as viewing distance decreases. If one assumes that the visual system knows how reliable each estimator is, one would predict that greater weight is given to the HSR and VSR estimate at near than at far distances, and this is in fact the case [7].

Whether each estimate is separately computed in its own neural process, and then given a weight, is not known. A maximum *a posteriori* Bayesian scheme that simply estimates the most likely slant given the observed signals behaves in a similar fashion as the weighted estimates model, though actual likelihood density (probability per deg of slant) is extraordinarily small in the case of stimuli that contain large cue conflicts [9]. The real visual system does not flinch, but instead produces a slant estimate that looks for all the world like a weighted average. It remains a possibility therefore that optimal slant estimation is implemented as a weighted combination of separate estimates.

We have now developed the theory to explain why HSR and VSR trade with each other at the "constancy" level of a single estimator (Equation 1), and why natural stimuli might appear the same as cue conflict stimuli (weighted averaging of estimates derived from exploitation of Equations 1 and 2, respectively). We next describe experiments that tested whether magnified (cue conflict) stimuli are distinguishable from natural (concordant) stimuli.

## 3   Existence of stereoscopic metamers

Stimuli were sparse random dot stereograms (RDS) on a black background, 28 deg in diameter, presented directly in front of the head using a haploscope. Observers performed a forced choice task with stimuli that contained different amounts of unilateral vertical and horizontal magnification. Vertical magnification was zero for the "A" stimuli, and 2% in the right eye for the "B" stimuli (1% minification in the left eye and 1% magnification in the right eye). Horizontal magnification was set at the value that nulled apparent slant in "A" stimuli (i.e. approximately 0%), and took on a range of values in "B" stimuli. Each trial consisted of two "A" stimuli and one "B" stimulus. The observer's task was to determine whether the three stimuli were presented in AAB or BAA order [13], i.e., whether the stimulus with vertical magnification was first or last of the three stimuli. Each stimulus was presented for 0.5 sec. Each stimulus was generated using a fresh set of 200 randomly positioned

dots. Each dot had a circular raised cosine luminance profile that was 30 arcmin in diameter. Three observers participated, including the author. Results are shown in Figure 1.

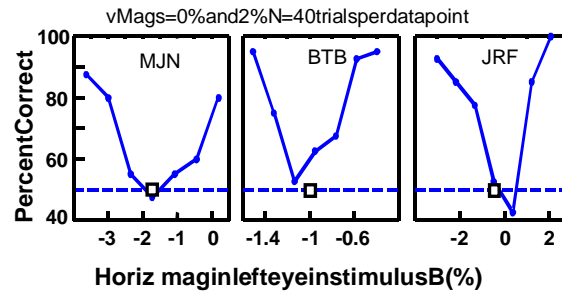

Figure 1. Observers are unable to distinguish 0% and 2% unilateral vertical magnification when unilateral horizontal magnification is added as well. Open squares show the horizontal magnification that evoked zero perceived slant under 2% vertical magnification.

For each observer, there was a value of horizontal magnification that, when added to the "B" stimulus, rendered it indistinguishable from the "A" stimulus. This is shown in Figure 1 by the fact that performance drops to chance (50%) at some value of horizontal magnification. From this experiment it is evident that stimuli with very different disparity patterns can be made perceptually indistinguishable in a forced-choice task with well-practiced observers.

### 3.1    Experimental conditions necessary for stereometamers

Several properties of the experiment were essential to the effect. First, the vertical magnification must not be to large. At large vertical magnifications it is still possible to null apparent slant, but the stimuli are distinguishable because the dots themselves look different (they look as though blurred in the vertical direction). Two out of three observers were able to distinguish the "A" and "B" stimuli 100% of the time when the vertical magnification was increased from 2% to 5%. Second, observers must be instructed to maintain fixation. If left and right saccades are allowed, the "B" stimulus appears slanted in the direction predicted by its horizontal magnification. This is a rather striking effect—the surface appears to change slant simply because one starts looking about. This effect was not found previously [14] but is predicted as a consequence of sequential stereopsis [15]. Finally, if the stimuli are shown for more than about 1 sec it is possible to distinguish "A" and "B" stimuli by making vertical saccades from the top to the bottom of the stimulus, by taking advantage of the fact that in forward gaze, vertical saccades have equal amplitude in the two eyes [16]. For "B" stimuli only, the dots are diplopic (seen in double vision) immediately after a saccade to the top (or bottom) of the stimulus. An automatic vertical vergence eye movement then brings the dots into register after about 0.5 sec. At that point a saccade to the bottom (or top) of the stimulus again causes diplopia.

## 4    Breaking metamerization though change of vergence eye posture

In the haploscope it was possible to present unchanged retinal images across a range of vergence eye postures. Stimuli that were metameric to each other with the eyes verged at 100 cm were presented again with the eyes verged at 20 cm. For three out

of four observers, the images were then distinguishable. Figure 2 illustrates this effect schematically, and Figure 3 quantifies it by plotting the amount of horizontal magnification that was needed to null apparent slant at each of the two vergence angles for one observer (left panel) and all four observers (right panel).

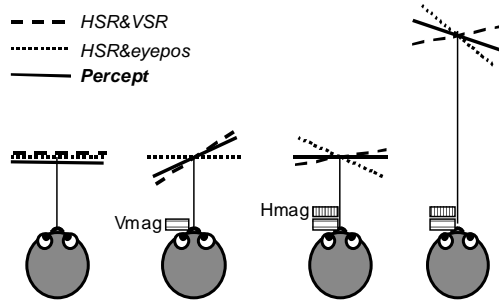

Figure 2. Schematic illustration of the effect of distance in the slant-nulling task. First panel: both stereoscopic methods of estimating slant indicate that the surface is frontoparallel, and it appears so. Second panel: a vertical magnifier is placed before one eye, changing the estimate that uses vertical disparity, but not the estimate that uses eye position. The resulting percept is a weighted average of the two. Third panel: horizontal magnification is added until the surface appears frontoparallel again. At this point the two stereo estimates have opposite sign. Fourth panel: increasing the apparent distance to the stimulus (by decreasing the vergence) scales up both estimates by the same factor. The surface no longer appears frontoparallel because the weighting of the estimates has changed.

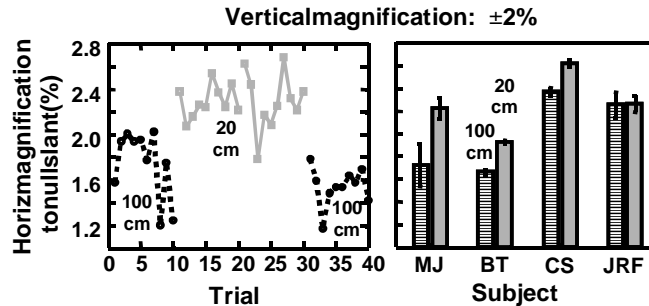

Figure 3. When the eyes were verged at 100 or 20 cm distance, different amounts of horizontal magnification were needed to null the slant induced by vertical magnification. Left: 10 settings that nulled slant at 100 cm, followed by 20 settings at 20 cm, followed by 10 at 100 cm (observer BTB). Right: three out of four observers show an effect of vergence *per se*. Error bars are SEs of the mean.

## 5 Comparison of perceptual and sensory metamers

The stimuli described here appear the same as a result of perceptual computations that occur well after transduction of light energy by the photoreceptors. Physically different stimuli that are transduced identically might be dubbed *sensory metamers.* One example of a sensory metamer is given by the trade between intensity and duration for briefly flashed lights (Bloch's Law [17]): two flashes containing the

same number of photons are indistinguishable if their durations are both less than 10 msec. Another example of sensory metamerization, that we will now consider in greater detail, is the traditional color metamer. The three cone photoreceptor types can support color vision because they are sensitive to different wavelengths of light. However, each cone type responds to a range of wavelengths, and two lights with different spectra may activate the three cone types identically. From that point on, the lights will be indistinguishable within the nervous system. (See [18] for a review of color metamers).

Table 1 summarizes several properties of color metamers, and analogous properties of our new stereo metamers. We can approximate the visible spectrum of a light by sampling its power within N different wavelength intervals, where N is large. Thus light $t$ can be represented by an Nx1 vector. Light $t'$ is metameric to $t$ if $Bt'=Bt$, where $B$ is the 3xN matrix whose rows represent the spectral sensitivities of the three cone mechanisms [19]. The transformation that maps one *stereo* metamer to another is simply a scaling of one eyes' image in the vertical and horizontal directions, with less scaling typically needed in the horizontal than vertical direction. Let u and v represent the x and y disparity, respectively, so that [u v] is a function of location (x,y) within the cyclopean image. Then two random-dot image pairs (representing flat surfaces slanted about a vertical axis) will be metameric if their disparity patterns, [u' v'] and [u v], are related to each other by [u' v'] = [u(1+$m$)v(1+ $n$)], where $m$ and $n$ are small (on the order of 0.01), with $m/n$ equal to the weight of $S_{HSR,VSR}$ in the final slant estimate.

Table 1: properties of color and stereo metamers

| PROPERTY | COLOR METAMERS | STEREO METAMERS |
|---|---|---|
| Metamer type: | Sensory | Perceptual |
| Site of loss: | Peripheral | Central (two places) |
| Loss process: | Transduction | Computation |
| Metameric class formation: | Lights $t'$ and $t$ are metameric iff $Bt'=Bt$, where $B$ is the 3xN matrix of cone spectral sensitivities | Disparity map [uv] is metameric to [u' v'] iff [u'v']=[u(1+ $m$)v(1+ $n$)] where $m$ and $n$ are small and in the proper ratio |
| Dimensionality reduction: | $N \rightarrow 3$ | loss of 1 degree of freedom |
| Etiology: | Capacity limit | Recovery of scene parameter |

Computation of surface slant removes one dimension from the set of all physical stimuli. Depending how the problem is framed, this is a reduction from 2 dimensions (HSR and VSR) to one (slant), or from many dimensions (all physical stimuli that represent slanted surfaces) to one fewer dimensions.

While color and stereo metamers can be described as sensory and perceptual, respectively, the boundary between these categories is fuzzy, as is the boundary between sensation and perception. Would motion metamers based on "early" motion detectors be sensory or perceptual? What of stimuli that look identical to retinal ganglion cells, after evoking different patterns of photoreceptor activity? While there is a real distinction to be made between sensory and perceptual metamers, but not all metamers need be easily categorized as one or the other.

## 5.1   The metamer hierarchy

Loftus[20] makes a distinction reminiscent of the one made here, between "memory metamers" and "perceptual metamers," with memory metamers being stimuli that evoke distinguishable percepts during live viewing, but that become indistinguishable after mnemonic encoding. Thus, Loftus classified as "perceptual" both our perceptual and sensory metamers. Figure 4 suggests how the three concepts are related. In this framework, color and stereo metamers are both perceptual metamers, but only color metamers are sensory metamers.

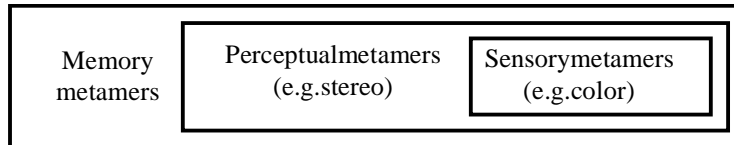

Figure 5. The metamer hierarchy.

# 6   Conclusions

At each vergence eye posture it was possible to create stereoscopic stimuli with distinct disparity patterns that were nonetheless indistinguishable in a forced choice task. Stimuli that were metamers with the eyes in one position became distinguishable after a change of vergence eye posture alone, without changes to the retinal images. We can conclude that horizontal disparity *per se* is lost to the visual system after combination with the other signals that are used to interpret it as depth. Presumably, stereo metamers have distinguishable representations in primary visual cortex—one suspects this would be evident in evoked potentials or fMRI. The loss of information that renders these stimuli metameric probably occurs in two places. First, there appears to be a leak-proof "constancy" computation in which vertical disparity is used to correct horizontal disparity (Equation 1). The output of this computation is unaffected if equal amounts of horizontal and vertical magnification are added to one eyes' image. However, the estimator that uses felt eye position can distinguish these stimuli, because their horizontal size ratios differ. Thus a second leak-proof step must occur, in which slant estimates are combined in a weighted average. It seems reasonable to call these stimuli "perceptual metamers," by analogy with, and to distinguish them from, the traditional "sensory" metamerization of colored lights.

### Acknowledgments

This work was supported by startup funds provided to the author by the University of Pennsylvania. The author thanks Mark Nolt for help conducting the experiments, Rufus Frazer for serving as an observer, and Jack Nachmias and David Brainard for comments on an earlier draft of this paper.

### References

1.   Clark, J.J. and A.L. Yuille, *Data fusion for sensory information processing systems*. 1990, Boston: Kluwer.

2.   Landy, M.S., *et al.*, *Measurement and modeling of depth cue combination: in defense of weak fusion.* Vision Research, 1995. **35**(3): p. 389-412.

3.   Ogle, K.N., *Induced size effect. I. A new phenomenon in binocular space perception associated with the relative sizes of the images of the two eyes.* Archives of Ophthalmology, 1938. **20**: p. 604-623.

4. Gårding, J., *et al.*, *Stereopsis, vertical disparity and relief transformations.* Vision Res, 1995. **35**(5):p.703-22.

5. Backus, B.T., *et al.*, *Horizontal and vertical disparity, eye position, and stereoscopic slant perception.* Vision Res, 1999. **39**(6):p.1143-70.

6. Banks, M.S. and B.T. Backus, *Extra-retinal and perspective cues cause the small range of the induced effect.* Vision Res, 1998. **38**(2):p.187-94.

7. Backus, B.T. and M.S. Banks, *Estimator reliability and distance scaling in stereoscopic slant perception.* Perception, 1999. **28**(2):p.217-42.

8. Foley, J.M., *Binocular distance perception.* Psychol Rev, 1980. **87**(5):p.411-34.

9. Backus, B.T. and M.J. Nolt, *Analysis of stereoscopic metamers.* Journal of Vision (Vision Sciences conference supplement), 2001. **1**:p. in press.

10. Stevens, K.A., *Slant-tilt: the visual encoding of surface orientation.* Biol Cybern, 1983. **46**(3):p.183-95.

11. Rogers, B.J. and M.F. Bradshaw, *Vertical disparities, differential perspective and binocular stereopsis.* Nature, 1993. **361**(6409):p.253-5.

12. Mayhew, J.E. and H. Longuet-Higgins, C, *A computational model of binocular depth perception.* Nature, 1982. **297**(5865):p.376-378.

13. Calkins, D.J., J.E. Thornton, and E.N. Pugh, Jr., *Monochromatism determined at a long-wavelength/middle-wavelength cone- antagonistic locus.* Vision Res, 1992. **32**(12):p.2349-67.

14. van Ee, R. and C.J. Erkelens, *Temporal aspects of binocular slant perception.* Vision Res, 1996. **36**(1):p.43-51.

15. Enright, J.T., *Sequential stereopsis: a simple demonstration.* Vision Res, 1996. **36**(2):p.307-12.

16. Schor, C.M., J. Gleason, and D. Horner, *Selective nonconjugate binocular adaptation of vertical saccades and pursuits.* Vision Res, 1990. **30**(11):p.1827-44.

17. Barlow, H.B., *Temporal and spatial summation in human vision at different backgound intensities.* Journal of Physiology, 1958. **141**:p.337-350.

18. Wandell, B.A., *Foundations of vision.* 1995, Sunderland, MA: Sinauer Associates.

19. Baylor, D.A., B.J. Nunn, and J.L. Schnapf, *Spectral sensitivity of cones of the monkey Macaca fascicularis.* J Physiol, 1987. **390**:p.145-60.

20. Loftus, G.R. and E. Ruthruff, *A theory of visual information acquisition and visual memory with special application to intensity-duration trade-offs.* J Exp Psychol Hum Percept Perform, 1994. **20**(1):p.33-49.
